# Learning nonlinear overcomplete representations for efficient coding

**Michael S. Lewicki**
lewicki@salk.edu

**Terrence J. Sejnowski**
terry@salk.edu

Howard Hughes Medical Institute
Computational Neurobiology Lab
The Salk Institute
10010 N. Torrey Pines Rd.
La Jolla, CA 92037

## Abstract

We derive a learning algorithm for inferring an overcomplete basis by viewing it as probabilistic model of the observed data. Overcomplete bases allow for better approximation of the underlying statistical density. Using a Laplacian prior on the basis coefficients removes redundancy and leads to representations that are sparse and are a *nonlinear* function of the data. This can be viewed as a generalization of the technique of independent component analysis and provides a method for blind source separation of fewer mixtures than sources. We demonstrate the utility of overcomplete representations on natural speech and show that compared to the traditional Fourier basis the inferred representations potentially have much greater coding efficiency.

A traditional way to represent real-values signals is with Fourier or wavelet bases. A disadvantage of these bases, however, is that they are not specialized for any particular dataset. Principal component analysis (PCA) provides one means for finding an basis that is adapted for a dataset, but the basis vectors are restricted to be orthogonal. An extension of PCA called independent component analysis (Jutten and Herault, 1991; Comon et al., 1991; Bell and Sejnowski, 1995) allows the learning of non-orthogonal bases. All of these bases are complete in the sense that they span the input space, but they are limited in terms of how well they can approximate the dataset's statistical density.

Representations that are overcomplete, *i.e.* more basis vectors than input variables, can provide a better representation, because the basis vectors can be specialized for

a larger variety of features present in the entire ensemble of data. A criticism of overcomplete representations is that they are redundant, *i.e.* a given data point may have many possible representations, but this redundancy is removed by the prior probability of the basis coefficients which specifies the *probability* of the alternative representations.

Most of the overcomplete bases used in the literature are fixed in the sense that they are not adapted to the structure in the data. Recently Olshausen and Field (1996) presented an algorithm that allows an overcomplete basis to be learned. This algorithm relied on an approximation to the desired probabilistic objective that had several drawbacks, including tendency to breakdown in the case of low noise levels and when learning bases with higher degrees of overcompleteness. In this paper, we present an improved approximation to the desired probabilistic objective and show that this leads to a simple and robust algorithm for learning optimal overcomplete bases.

# 1   Inferring the representation

The data, $x_{1:L}$, are modeled with an overcomplete linear basis plus additive noise:

$$\mathbf{x} = \mathbf{A}\mathbf{s} + \epsilon \tag{1}$$

where $\mathbf{A}$ is an $L \times M$ matrix, whose columns are the basis vectors, where $M \geq L$. We assume Gaussian additive noise so that $\log P(\mathbf{x}|\mathbf{A}, \mathbf{s}) \propto -\lambda(\mathbf{x} - \mathbf{A}\mathbf{s})^2/2$, where $\lambda = 1/\sigma^2$ defines the precision of the noise.

The redundancy in the overcomplete representation is removed by defining a density for the basis coefficients, $P(\mathbf{s})$, which specifies the *probability* of the alternative representations. The most probable representation, $\hat{\mathbf{s}}$, is found by maximizing the posterior distribution

$$\hat{\mathbf{s}} = \max_{\mathbf{s}} P(\mathbf{s}|\mathbf{A}, \mathbf{x}) = \max_{\mathbf{s}} P(\mathbf{s})P(\mathbf{x}|\mathbf{A}, \mathbf{s}) \tag{2}$$

$P(\mathbf{s})$ influences how the data are fit in the presence of noise and determines the uniqueness of the representation. In this model, the data is a linear function of $\mathbf{s}$, but $\mathbf{s}$ is *not*, in general, a linear function of the data. If the basis function is complete ($\mathbf{A}$ is invertible) then, assuming broad priors and low noise, the most probable internal state can be computed simply by inverting $\mathbf{A}$. In the case of an overcomplete basis, however, $\mathbf{A}$ can not be inverted. Figure 1 shows how different priors induce different representations.

Unlike the Gaussian prior, the optimal representation under the Laplacian prior cannot be obtained by a simple linear operation. One approach for optimizing $\mathbf{s}$ is to use the gradient of the log posterior in an optimization algorithm. An alternative method for finding the most probable internal state is to view the problem as the linear program: $\min \mathbf{1}^T \mathbf{s}$ such that $\mathbf{A}\mathbf{s} = \mathbf{x}$. This can be generalized to handle both positive and negative $\mathbf{s}$ and solved efficiently and exactly with interior point linear programming methods (Chen et al., 1996).

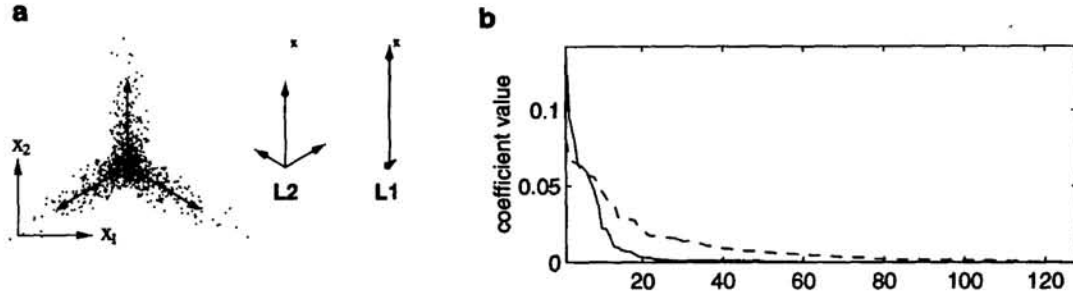

Figure 1: Different priors induce different representations. (a) The 2D data distribution has three main axes which form an overcomplete representation. The graphs marked "L2" and "L1" show the optimal scaled basis vectors for the data point x under the Gaussian and Laplacian prior, respectively. Assuming zero noise, a Gaussian for $P(\mathbf{s})$ is equivalent to finding the exact fitting s with minimum $L_2$ norm, which is given by the pseudoinverse $\mathbf{s} = \mathbf{A}^+\mathbf{x}$. A Laplacian prior ($P(s_m) \propto \exp[-\theta|s_m|]$) yields the exact fit with minimum $L_1$ norm, which is a *nonlinear* operation which essentially selects a subset of the basis vectors to represent the data (Chen et al., 1996). The resulting representation is sparse. (b) A 64-sample segment of speech was fit to a 2× overcomplete Fourier representation (128 basis vectors). The plot shows rank order distribution of the coefficients of s under a Gaussian prior (dashed); and a Laplacian prior (solid). Far more significantly positive coefficients are required under the Gaussian prior than under the Laplacian prior.

## 2   Learning

The learning objective is to adapt **A** to maximize the probability of the data which is computed by marginalizing over the internal states

$$P(\mathbf{x}|\mathbf{A}) = \int ds\, P(\mathbf{s})P(\mathbf{x}|\mathbf{A}, \mathbf{s}) \tag{3}$$

general, this integral cannot be evaluated analytically but can be approximated with a Gaussian integral around $\hat{\mathbf{s}}$, yielding

$$\log P(\mathbf{x}|\mathbf{A}) \approx \text{const.} + \log P(\hat{\mathbf{s}}) - \frac{\lambda}{2}(\mathbf{x} - \mathbf{A}\hat{\mathbf{s}})^2 - \frac{1}{2}\log \det \mathbf{H} \tag{4}$$

where **H** is the Hessian of the log posterior at $\hat{\mathbf{s}}$, given by $\lambda\mathbf{A}^{\mathrm{T}}\mathbf{A} - \nabla\nabla \log P(\hat{\mathbf{s}})$. To avoid a singularity under the Laplacian prior, we use the approximation $(\log P(s_m))' \approx -\theta \tanh(\beta s_m)$ which gives the Hessian full rank and positive determinant. For large $\beta$ this approximates the true Laplacian prior. A learning rule can be obtained by differentiating $\log P(\mathbf{x}|\mathbf{A})$ with respect to **A**.

In the following discussion, we will present the derivations of the three terms in (4) and simplifying assumptions that lead to the following simple form of the learning rule

$$\Delta\mathbf{A} = \mathbf{A}\mathbf{A}^{\mathrm{T}}\nabla \log P(\mathbf{x}|\mathbf{A}) \approx -\mathbf{A}(\mathbf{z}\hat{\mathbf{s}}^{\mathrm{T}} + \mathbf{I}) \tag{5}$$

where $z_k = \partial \log P(s_k)/\partial s_k$.

## 2.1  Deriving $\nabla \log P(\hat{\mathbf{s}})$

This term specifies how to change $\mathbf{A}$ so as to make the probability of the representation $\hat{\mathbf{s}}$ more probable. If we assume a Laplacian prior, this component changes $\mathbf{A}$ to make the representation more sparse.

We assume $P(\hat{\mathbf{s}}) = \prod_m P(\hat{s}_m)$. In order to obtain $\partial \hat{s}_m / \partial a_{ij}$, we need to describe $\hat{\mathbf{s}}$ as a function of $\mathbf{A}$. If the basis is complete (and we assume low noise), then we may simply invert $\mathbf{A}$ to obtain $\hat{\mathbf{s}} = \mathbf{A}^{-1}\mathbf{x}$. When $\mathbf{A}$ is overcomplete, however, there is no simple expression, but we may still make an approximation.

Under priors, the most probable solution, $\hat{\mathbf{s}}$, will yield at most $L$ non-zero elements. In effect, this selects a complete basis from $\mathbf{A}$. Let $\check{\mathbf{A}}$ represent this reduced basis under $\hat{\mathbf{s}}$. We then have $\check{\mathbf{s}} = \check{\mathbf{A}}^{-1}(\mathbf{x} - \epsilon)$ where $\check{\mathbf{s}}$ is equal to $\hat{\mathbf{s}}$ with $M - L$ zero-valued elements removed. $\check{\mathbf{A}}^{-1}$ obtained by removing the columns of $\mathbf{A}$ corresponding to the $M - L$ zero-valued elements of $\hat{\mathbf{s}}$. This allows us to use results obtained for the case when $\mathbf{A}$ is invertible. Following MacKay (1996) we obtain

$$\frac{\partial \check{s}_k}{\partial \check{a}_{ij}} = -\sum_l \check{A}_{ki}^{-1} \check{A}_{jl}^{-1}(x_l - \epsilon_l) = -\check{A}_{ki}^{-1}\check{s}_j \tag{6}$$

Rewriting in matrix notation we have

$$\frac{\partial \log P(\check{\mathbf{s}})}{\partial \mathbf{A}} = -\check{\mathbf{A}}^{-T}\check{\mathbf{z}}\check{\mathbf{s}}^T \tag{7}$$

We can use this to obtain an expression in terms of the original variables. We simply invert the mapping $\hat{\mathbf{s}} \to \check{\mathbf{s}}$ to obtain $\mathbf{z} \leftarrow \check{\mathbf{z}}$ and $\mathbf{W}^T \leftarrow \check{\mathbf{A}}^{-T}$ (row-wise) with $z_m = 0$ and row $m$ of $\mathbf{W}^T = 0$ if $\check{s}_m = 0$. We then have

$$\frac{\partial \log P(\mathbf{s})}{\partial \mathbf{A}} = -\mathbf{W}^T\mathbf{z}\mathbf{s}^T \tag{8}$$

## 2.2  Deriving $\nabla(\mathbf{x} - \mathbf{A}\hat{\mathbf{s}})^2$

The second term specifies how to change $\mathbf{A}$ so as to minimize the data misfit. Letting $e_k = [\mathbf{x} - \mathbf{As}]_k$ and using the results and notation from above we have:

$$\frac{\partial}{\partial a_{ij}} \frac{\lambda}{2} \sum_k e_k^2 = \lambda e_i s_j + \lambda \sum_k e_k \sum_l a_{kl} \frac{\partial s_l}{\partial a_{ij}} \tag{9}$$

$$= \lambda e_i s_j + \lambda \sum_k e_k \sum_l -a_{kl} w_{li} s_j \tag{10}$$

$$= \lambda e_i s_j - \lambda e_i s_j = 0 \tag{11}$$

Thus no gradient component arises from the error term.

## 2.3  Deriving $\nabla \log \det \mathbf{H}$

The third term in the learning rule specifies how to change the weights so as to minimize the width of the posterior distribution $P(\mathbf{x}|\mathbf{A})$ and thus increase the overall probability of the data. An element of $\mathbf{H}$ is defined by $\mathbf{H}_{mn} = c_{mn} + b_{mn}$

where $c_{mn} = \sum_k \lambda a_{km} a_{kn}$ and $b_{mn} = [-\nabla\nabla \log P(\hat{s})]_{mn}$. This gives

$$\frac{\partial \log \det \mathbf{H}}{\partial a_{ij}} = \sum_{mn} \mathbf{H}_{nm}^{-1} \left[ \frac{\partial c_{mn}}{\partial a_{ij}} + \frac{\partial b_{mn}}{\partial a_{ij}} \right] \tag{12}$$

First considering $\partial c_{mn}/\partial a_{ij}$, we can obtain

$$\sum_{mn} \mathbf{H}_{mn}^{-1} \frac{\partial c_{mn}}{\partial a_{ij}} = \sum_{m \neq j} \mathbf{H}_{mj}^{-1} \lambda a_{im} + \sum_{m \neq j} \mathbf{H}_{jm}^{-1} \lambda a_{im} + \mathbf{H}_{jj}^{-1} 2\lambda a_{ij} \tag{13}$$

Using the fact that $\mathbf{H}_{mj}^{-1} = \mathbf{H}_{jm}^{-1}$ due to the symmetry of the Hessian, we have

$$\sum_{mn} \mathbf{H}_{mn}^{-1} \frac{\partial c_{mn}}{\partial \mathbf{A}} = 2\lambda \mathbf{A} \mathbf{H}^{-1} \tag{14}$$

Next we derive $\partial b_{mn}/\partial a_{ij}$. We have that $\nabla\nabla \log P(\hat{s})$ is diagonal, because we assume $P(\hat{s}) = \prod_m P(\hat{s}_m)$. Letting $2y_m = \mathbf{H}_{mm}^{-1} \partial b_{mm}/\partial \hat{s}_m$ and using the result under the reduced representation (6) we can obtain

$$\sum_{mm} \mathbf{H}_{mm}^{-1} \frac{\partial b_{mm}}{\partial \mathbf{A}} = -2\mathbf{W}^{\mathrm{T}} \mathbf{y}\hat{\mathbf{s}}^{\mathrm{T}} \tag{15}$$

## 2.4 Stabilizing and simplifying the learning rule

Putting the terms together yields a problematic expression due to the matrix inverses. This can be alleviated by multiplying the gradient by an appropriate positive definite matrix, which rescales the gradient components but preserves a direction valid for optimization. Noting that $\mathbf{A}^{\mathrm{T}}\mathbf{W}^{\mathrm{T}} = \mathbf{I}$ we have

$$\mathbf{A}\mathbf{A}^{\mathrm{T}} \nabla \log P(\mathbf{x}|\mathbf{A}) = -\mathbf{A}\mathbf{z}\hat{\mathbf{s}}^{\mathrm{T}} - \lambda \mathbf{A}\mathbf{A}^{\mathrm{T}}\mathbf{A}\mathbf{H}^{-1} + \mathbf{A}\mathbf{y}\hat{\mathbf{s}}^{\mathrm{T}} \tag{16}$$

If $\lambda$ is large (low noise) then the Hessian is dominated by $\lambda \mathbf{A}^{\mathrm{T}}\mathbf{A}$ and we have

$$-\lambda \mathbf{A}\mathbf{A}^{\mathrm{T}}\mathbf{A}\mathbf{H}^{-1} = -\mathbf{A}\lambda\mathbf{A}^{\mathrm{T}}\mathbf{A}(\lambda\mathbf{A}^{\mathrm{T}}\mathbf{A} + \mathbf{B})^{-1} \approx -\mathbf{A} \tag{17}$$

The vector $\mathbf{y}$ hides a computation involving the inverse Hessian. If the basis vectors in $\mathbf{A}$ are randomly distributed, then as the dimensionality of $\mathbf{A}$ increases the basis vectors become approximately orthogonal and consequently the Hessian becomes approximately diagonal. It can be shown that if $\log P(\mathbf{s})$ and its derivatives are smooth, $y_m$ vanishes for large $\lambda$. Combining the remaining terms yields equation (5). Note that this rule contains no matrix inverses and the vector $\mathbf{z}$ involves only the derivative of the log prior.

In the case where $\mathbf{A}$ is square, this form of the rule is similar to the natural gradient independent component analysis (ICA) learning rule (Amari et al., 1996). The difference in the more general case where $\mathbf{A}$ is rectangular is that $\hat{s}$ must maximize the posterior distribution $P(\mathbf{s}|\mathbf{x}, \mathbf{A})$ which cannot be done simply with the filter matrix as in standard ICA algorithms.

## 3  Examples

**More sources than inputs.** In these 2D examples, the bases were initialized to random, normalized vectors. The coefficients were solved using BPMPD and publicly available interior point linear programming package (Meszaros, 1997) which gives the most probable solution under the Laplacian prior assuming zero noise. The algorithm was run for 30 iterations using equation (5) with a stepsize of 0.001 and a batchsize of 200. Convergence was rapid, typically requiring less than 20 iterations. In all cases, the direction of the learned vectors matched those of the true generating distribution; the magnitude was estimated less precisely, possibly due to the approximation of $\log P(\mathbf{x}|\mathbf{A})$. This can be viewed as a source separation problem, but true separation will be limited due to the projection of the sources down to a smaller subspace which necessarily loses information.

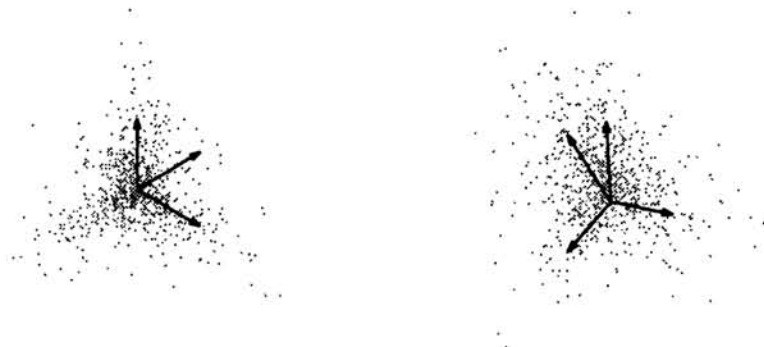

Figure 2: Examples illustrating the fitting of 2D distributions with overcomplete bases. The first example is equivalent to 3 sources mixed into 2 channels; the second to 4 sources mixed into 2 channels. The data in both examples were generated from the true basis $\mathbf{A}$ using $\mathbf{x} = \mathbf{As}$ with the elements of $\mathbf{s}$ distributed according to an exponential distribution with unit mean. Identical results were obtained by drawing $\mathbf{s}$ from a Laplacian prior (positive and negative coefficients). The overcomplete bases allow the model to capture the true underlying statistical structure in the 2D data space.

**Overcomplete representations of speech.** Speech data were obtained from the TIMIT database, using a single speaker was speaking ten different example sentences with no preprocessing. The basis was initialized to an overcomplete Fourier basis. A conjugate gradient routine was used to obtain the most probable basis coefficients. The stepsize was gradually reduced over 10000 iterations. Figure 3 shows that the learned basis is quite different from the Fourier representation. The power spectrum for the learned basis vectors can be multimodal and/or broadband. The learned basis achieves greater coding efficiency: $2.19 \pm 0.59$ bits per sample compared to $3.86 \pm 0.28$ bits per sample for a 2× overcomplete Fourier basis.

## 4  Summary

Learning overcomplete representations allows a basis to better approximate the underlying statistical density of the data and consequently the learned representations have better encoding and denoising properties than generic bases. Unlike the case for complete representations and the standard ICA algorithm, the transformation

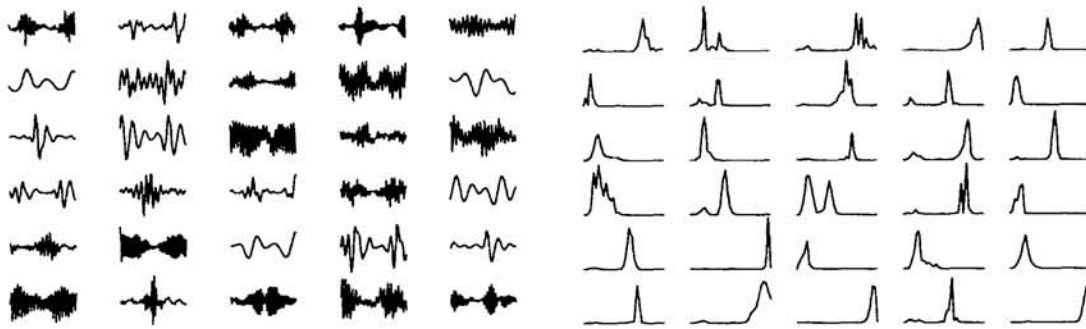

Figure 3: An example of fitting a 2x overcomplete representation to segments of from natural speech. Each segment consisted of 64 samples, sampled at a frequency of 8000 Hz (8 msecs). The plot shows a random sample of 30 of the 128 basis vectors (each scaled to full range). The right graph shows the corresponding power spectral densities (0 to 4000 Hz).

from the data to the internal representation is non-linear. The probabilistic formulation of the basis inference problem offers the advantages that assumptions about the prior distribution on the basis coefficients are made explicit and that different models can be compared objectively using $\log P(\mathbf{x}|\mathbf{A})$.

# References

Amari, S., Cichocki, A., and Yang, H. H. (1996). A new learning algorithm for blind signal separation. In *Advances in Neural and Information Processing Systems*, volume 8, pages 757–763, San Mateo. Morgan Kaufmann.

Bell, A. J. and Sejnowski, T. J. (1995). An information maximization approach to blind separation and blind deconvolution. *Neural Computation*, 7(6):1129–1159.

Chen, S., Donoho, D. L., and Saunders, M. A. (1996). Atomic decomposition by basis pursuit. Technical report, Dept. Stat., Stanford Univ., Stanford, CA.

Comon, P., Jutten, C., and Herault, J. (1991). Blind separation of sources .2. problems statement. *Signal Processing*, 24(1):11–20.

Jutten, C. and Herault, J. (1991). Blind separation of sources .1. an adaptive algorithm based on neuromimetic architecture. *Signal Processing*, 24(1):1–10.

MacKay, D. J. C. (1996). Maximum likelihood and covariant algorithms for independent component analysis. University of Cambridge, Cavendish Laboratory. Available at `ftp://wol.ra.phy.cam.ac.uk/pub/mackay/ica.ps.gz`.

Meszaros, C. (1997). BPMPD: An interior point linear programming solver. Code available at `ftp://ftp.netlib.org/opt/bpmpd.tar.gz`.

Olshausen, B. A. and Field, D. J. (1996). Emergence of simple-cell receptive-field properties by learning a sparse code for natural images. *Nature*, 381:607–609.
